# Maximal Cliques that Satisfy Hard Constraints with Application to Deformable Object Model Learning

**Xinggang Wang**[1]* **Xiang Bai**[1] **Xingwei Yang**[2]† **Wenyu Liu**[1] **Longin Jan Latecki**[3]

[1] Dept. of Electronics and Information Engineering, Huazhong Univ. of Science and Technology, China
[2] Image Analytics Lab, GE Research, One Research Circle, Niskayuna, NY 12309, USA
[3] Dept. of Computer and Information Sciences, Temple Univ., USA

{wxghust,xiang.bai}@gmail.com,yang@ge.com,liuwy@hust.edu.cn,latecki@temple.edu

## Abstract

We propose a novel inference framework for finding maximal cliques in a weighted graph that satisfy hard constraints. The constraints specify the graph nodes that must belong to the solution as well as mutual exclusions of graph nodes, i.e., sets of nodes that cannot belong to the same solution. The proposed inference is based on a novel particle filter algorithm with state permeations. We apply the inference framework to a challenging problem of learning part-based, deformable object models. Two core problems in the learning framework, matching of image patches and finding salient parts, are formulated as two instances of the problem of finding maximal cliques with hard constraints. Our learning framework yields discriminative part based object models that achieve very good detection rate, and outperform other methods on object classes with large deformation.

## 1 Introduction

The problem of finding maximal cliques in a weighted graph is faced in many applications from computer vision to social networks. Related work on finding dense subgraph in weighted graph include [16, 12, 14]. However, these approaches relax the discrete problem of subgraph selection to a continuous problem. The main drawback of such relaxation is the fact that it is impossible to enforce that the constraints are satisfied for solutions of the relaxed problem. Therefore, we aim at solving the discrete subgraph selection problem by employing the recently proposed extension of particle filter inference to problems with state permeations [20]. There are at least two main contributions of this paper: (1) We propose an inference framework for solving a maximal clique problem that cannot be solved with typical clustering methods nor with recent relaxation based methods [16, 12, 14]. (2) We utilize the inference framework for solving a challenging problem of learning a part model for deformable object detection.

Object detection is one of the key challenges in computer vision, due to the large intra-class appearance variation of an object class. The appearance variation arises not only from changes in illumination, viewpoint, color, and other visual properties, but also from nonrigid deformations. Objects under deformation often observed large variation globally. However, their local structures are somewhat more invariant to the deformations. Based on this observation, we propose a learning by matching framework to match all local image patches from training image. By matching, object parts with similar local structure in different training images can be found.

Given a set of training images that contain objects of the same class, e.g., Fig. 1(a), our first problem is to select a set of image patches that depict the same visual part of these objects. Thus, an object part is regarded as a collection of image patches e.g., Fig. 1(c). To solve the problem, we divide each training image into a set of overlapping patches, like the ones shown in Fig. 1(b), and construct a graph whose nodes represent the patches. The edge weights represent the appearance similarity of pair of patches. Since close by patches in the same image tend to be very similar, we must impose

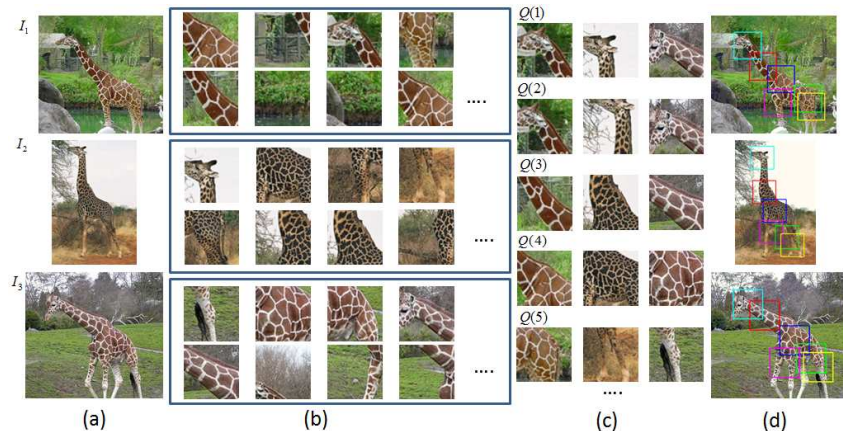

Figure 1: (a) example training images; (b) patches extracted from the training images; (c) object parts as collections of patches obtained as maximal cliques of patch similarity graph; (d) the learned salient parts for giraffe, the patches belong to the same salient part are in the same color. The salient parts are obtained as maximal cliques in a second graph whose vertices represent the object parts.

a hard constraint that a patch set representing the same object part does not contain two patches from the same image. This constraint is very important, since otherwise very similar patches from the same images will dominate this graph. In order to obtain meaningful object parts, we define an object part as a maximal clique in the weighted graph that satisfies the above constraint. By solving the problem of maximal clique, we obtain a set of object parts like the ones shown in Fig. 1(c). We use this set as vertices of a second graph. Finally, we obtain a small set of salient visual parts, e.g., Fig. 1(d), by solving a different instance of the maximal clique problem on the second graph.

For each salient visual part, we train a discriminative classifier. By combining these classifiers with spatial distribution of the salient object parts, a detector for deformable object is built. As illustrated in the experimental results, this detector achieves very good object detection performance, and outperforms other methods on object classes with large deformation.

The computer vision literature has approached learning of part based object models in different ways. In [8] objects are modeled as flexible constellations of parts, parts are constrained to a sparse set of locations determined by an entropy-based feature detector, other part models based on feature detector include [15, 17]. Our model is similar to discriminatively trained part based model in [6] in that we train SVM classifiers for each part of object and geometric arrangement of parts is captured by a set of "springs". However, our learning method is quite different from [6]. In [6] the learning problem is formalized as latent SVM, where positions of parts are considered as latent values. The learning process is an iterative algorithm that alternates between fixing latent values and optimizing the latent SVM objective function. In contrast, we case part learning as finding maximal cliques in a weighted graph of image patches. The edge weights represent appearance similarities of patches. In [4, 13] multiple instance learning is used to search position of object parts in training images, and boosting algorithm is used to select salient parts to represent object.

## 2 Maximal Cliques that Satisfy Hard Constraint

A weighted graph $G$ is defined as $G = (V, E, e)$, where $V = \{v_1, \ldots, v_n\}$ is the vertex set, $n$ is the number of vertices, $E \subseteq V \times V$, and $e : E \to \mathbb{R}_{\geq 0}$ is the weight function. Vertices in $G$ correspond to data points, edge weights between different vertices represent the strength of their relationships, and self-edge weight respects importance of a vertex. As is customary, we represent the graph $G$ with the corresponding weighted adjacency matrix, more specifically, an $n \times n$ symmetric matrix $A = (a_{ij})$, where $a_{ij} = e(v_i, v_j)$ if $(v_i, v_j) \in E$, and $a_{ij} = 0$ otherwise.

Let $S = \{1, ..., n\}$ be the index set of vertex set $V$. For any subset $T \subseteq S$, $G_T$ denotes a subgraph of $G$ with vertex set $V_T = \{v_i, i \in T\}$ and edge set $E_T = \{(v_i, v_j) \mid (v_i, v_j) \in E, i \in T, j \in T\}$. The total weight of subgraph $G_T$ is defined as $f(G_T) = \sum_{i \in T, j \in T} A(i, j)$. We can express $T$ by an indicator vector $x = (x_1, \ldots, x_n) \in \{0, 1\}^n$ such that $x_i = 1$ if $i \in T$ and $x_i = 0$ otherwise. Then $f(G_T)$ can be represented in a quadratic form $f(x) = x^T A x$.

We consider mutex relationship between vertices in graph. Given a subset of vertices $M \subseteq S$, we call $M$ a *mutex* (short for mutual exclusion) if $i \in M$ and $j \in M$ implies that vertices $v_i$ and $v_j$ can not belong to the same maximal clique. Formally, $M$ is a constraint on the indicator vector $x \in \{0,1\}^n$, i.e., if $i \in M$ and $j \in M$, then $x_i + x_j \leq 1$. A *mutex set* of graph $G$ is $\mathcal{M} = \{M_1, \ldots, M_m \mid M_i \subseteq S, i = 1, \ldots, m\}$ such that each $M_i$ is a mutex for $i = 1, \ldots, m$. Given a set $T \subseteq S$, we define mutex$(T)$ as a set of indices of vertices of $G$ that are incompatible with $T$ according to $\mathcal{M}$: mutex$(T) = \{j \in S | \exists_{M_i \in \mathcal{M}} \exists_{k \in T} j, k \in M_i\}$. We consider the following maximization problem

$$
\begin{aligned}
\underset{x}{\text{maximize}} \quad & f(x) = x^T A x \\
\text{subject to} \quad & \text{(C1) } x = (x_1, \ldots, x_n) \in \{0,1\}^n \text{ and} \\
& \text{(C2) } \forall i \in U \;\; x_i = 1 \text{ and} \\
& \text{(C3) } x_i + x_j \leq 1 \text{ if } \exists M_k \in \mathcal{M} \text{ such that } i, j \in M_k \text{ and} \\
& \text{(C4) } \Sigma x \leq K
\end{aligned}
\tag{1}
$$

The constraint (C2) specifies a set of vertices $U \subseteq S$ that must be selected as part of the solution, (C3) ensures that all mutex constraints are satisfied, (C4) requires number of vertices in the solution is small or equal to $K$. Of course, we assume the problem (1) is well-defined in that there exists $x$ that satisfies the four constraints (C1)-(C4).

The goal of (1) is to select a subset of vertices of graph $G$ such that $f$ is maximized and the constraints (C1)-(C4) are satisfied. Since $f$ is the sum of pairwise affinities of the elements of the selected subset, the larger is the subset, the larger is the value of $f$. However, the size of the subset is limited by the mutex constraints (C3) and maximal size constraint (C4).

A global maximum of (1) is called a $U - \mathcal{M}$ *maximal clique* of graph $G$. When both sets $U$ and $\mathcal{M}$ are clear from the context, we simply call the solution a maximal clique.

The problem (1) is a combinatorial optimization problem, and hence it is NP-hard [2]. As is the case for similar problems of finding dense subgraphs, the constraint (C1) is usually relaxed to $x \in [0,1]^n$, i.e., each coordinate of $x$ is relaxed to a continuous variable in the interval [0, 1], e.g., [16, 12, 14]. However, it is difficult if not impossible to ensure that constraints (C2), (C3) and (C4) are satisfied then. Another difficulty is related to discretization of the relaxed solution in order to obtain a solution that satisfies (C1). For these reasons, and since for our application, it is very important that the constraints are satisfied, we treat (C1)-(C4) as hard constraints that cannot be violated. We propose an efficient method for directly solving (1) in Section 4. We first present two instances of problem (1) in Section 3, where we describe the proposed application to learning salient object parts.

## 3 Learning by Matching

In this section, we present a novel framework to learn part based object model based on matching. The core problems of learning part based object model are how to search right locations of an object part in all training images and how to select salient parts for representing object. In our framework, the two problems are formulated as finding maximal cliques with hard constraints.

### 3.1 Matching Image Patches

Given a batch of training images $I = \{I_1, \ldots, I_K\}$ showing objects from a given class, e.g., Fig. 1 (a), where $K$ is the total number of training images. For every training image, we densely extract image patches with overlap. We denote the set of patches extracted from all images as $\{P_1, \ldots, P_n\}$, where $n$ is total number of patches. Each patch is described as $P_i = \{F_i, L_i, X_i, Y_i\}$ for $i \in [1, \ldots n]$, where $F_i$ is the appearance descriptor of $P_i$ (we use the descriptor from [19]), $L_i$ is the image label of $P_i$, (e.g., if $P_i$ is extracted from the 5th training image, $L_i = 5$), $X_i$ and $Y_i$ indicate the position of $P_i$ in its image. All the training images are normalized to the same size.

We treat all the patches as the set of vertices of graph $G$, i.e., $V = \{P_1, \ldots, P_n\}$. The affinity relation between the patches, i.e., the graph edge weights, are defined as $a_{ij} = F_i \cdot F_j$, if $i \neq j$, and $a_{ij} = 0$ otherwise, where $F_i \cdot F_j$ is the dot product of two feature vectors, which are normalized. It measures the appearance similarity of patches $P_i$ and $P_j$. In addition, if the distance between patch positions $(X_i, Y_i)$ and $(X_j, Y_j)$ is larger than 0.2 of the mean of all bounding box heights, we set $a_{ij} = 0$. This ensures that matrix A is sparse.

We have exactly $K$ mutex constraints $\mathcal{M} = \{M_1, \ldots, M_K\}$, where $M_j$ contains all patches from image $I_j$, i.e., $M_j = \{P_i \in V | L_i = j\}, j \in [1, \ldots, K]$. This means that we do not want two patches from the same image to belong to the same maximal clique.

Suppose that the first $r$ patches $P_1, \ldots, P_r$ are in the 1st training image, i.e., $L_i = 1$ if and only if $i = 1, \ldots, r$. The part learning algorithm by finding maximal cliques is given in Alg. 1.

---

**Algorithm 1** Part learning by finding maximal cliques with hard constraints

---

    **Input**: $A$, $\mathcal{M}$, $K$, and $r$.
    **for** $i = 1 \to r$ **do**
        1. Set $U = \{i\}$.
        2. Solve problem (1), get the solution $x^*$, and its value $W(i) = f(x^*) = x^{*T} A x^*$.
        3. Set the solution patches as $Q(i) = \{P_j | x_j^* = 1\}$.
    **end for**
    **Output**: Parts $\mathcal{Q} = \{Q(1), \ldots, Q(r)\}$ and their matching weights $\mathcal{W} = \{W(1), \ldots, W(r)\}$.

---

We recall that each learned part $Q(i)$ is defined as a set of $K$ patches, e.g., Fig. 1 (c). Due to our mutex constraint, each $Q(i)$ contains exactly one patch from each of $K$ training images. We treat the learned parts as candidate object parts, because there are non-object areas inside the bounding box images. Each value $W(i)$ represents a matching score of of $Q(i)$.

### 3.2 Selecting Salient Parts for Part Based Object Representation

In order to select a set of object parts that best represent the object class, our strategy is to find a subset of $\mathcal{Q}$ that maximizes the sum of the matching scores. We formulate this problem as finding maximal clique with hard constraints again. We define a new graph $H$ with vertices $V = \mathcal{Q}$ and adjacency matrix $B = (b_{ij})$, where $b_{ij} = W(i)$ if $i = j$, and $b_{ij} = 0$ otherwise. Thus, the matrix of graph $H$ has nonzero entries only on diagonal. It may appear that the problem is trivial, since there is no edges between different vertices of $H$, but this is not the case due to the mutex relations.

The mutex set $\mathcal{M}^H = \{M_1^H, \ldots, M_r^H\}$ is defined as $M_i^H = \{j \mid D(i, j) \leq \tau\}$ for $i, j \in [1, \ldots, r]$, where $\tau$ is a distance threshold and $D(i, j)$ is the average distance between patches in $Q(i)$ and $Q(j)$ that belong to the same image. If $Q(i)$ is selected as a salient part, the mutex $M_i^H$ ensures that the patches of other salient parts are not too close to the patches of $Q(i)$. For example, $Q(1)$ and $Q(2)$ in Fig. 1(c) both have good matching weights, but the average distance between $Q(1)$ and $Q(2)$ is smaller than $\tau$, so they cannot be selected as salient parts at the same time.

As initialization (C2), we set $U^H$ to a one element set containing $\arg\max_i W(i)$, so the part with maximal matching score is always selected as a salient part. We set $K$ in (C4) to $K^H$, where $K^H$ is the maximal number of salient parts. $K^H = 6$ in all our experiments.

By solving the second instance of problem (1) for $B$, $U^H$, $\mathcal{M}^H$, $K^H$, we obtain the set of salient parts as the solution $x^*$. We denote is as $\mathcal{SP} = \{Q(j) \mid x^*(j) = 1\}$.

## 4 Particle Filter Inference for $U - \mathcal{M}$ Maximal Clique

By associating a random variable (RV) $X_i$ with each vertex $i \in S$ of graph $G$, we introduce a Gibbs random field (GRF) with the neighborhood structure of graph $G$. Each RV can be assigned either 1 or 0, where $X_i = 1$ means that the vertex $v_i$ is selected as part of the solution. The probability of the assignment of values to all RVs is defined as

$$P(X_1 = x_1, \ldots, X_n = x_n) = p(x) \propto \exp \frac{f(x)}{\gamma} = \exp \frac{x^T A x}{\gamma}, \qquad (2)$$

where we recall that $x = (x_1, \ldots, x_n) \in \{0, 1\}^n$ and $\gamma > 0$. We observe that the definition in (2) also applies to a subset of RVs, i.e., we can use it to compute $P(X_{i_1} = x_{i_1}, \ldots, X_{i_k} = x_{i_k}) = p(x_{i_1}, \ldots, x_{i_k}) \propto \exp \frac{f(x_{i_1}, \ldots, x_{i_k})}{\gamma}$ for $k < n$. This is equivalent to setting other coordinates in the indicator vector $x$ to zero.

Since $\exp$ is a monotonically increasing function, the maximum of (2) is obtained at the same point as the maximum of $f$ in (1). We propose to utilize Particle Filter (PF) framework to maximize (2) subject to the constraints in (1). The goal of PF is to approximate $p(x)$ with a set of with weighted

samples $\{x^{(i)}, w(x^{(i)})\}_{i=1}^{N}$ drawn from some proposal distribution $q$. Under reasonable assumptions on $p(x)$ this approximation is possible with any precision if $N$ is sufficiently large [3].

Since it is still computationally intractable to draw samples from $q$ due to high dimensionality of $x$, PF utilizes Sequential Importance Sampling (SIS). In the classical PF approaches, samples are generated recursively following the order of the RVs according to $x_t^{(i)} \sim q(x_t|x_{1:t-1})$ for $t = 1, \ldots n$, and the particles are built sequentially $x_{1:t}^{(i)} = (x_{1:t-1}^{(i)}, x_t^{(i)})$ for $i = 1, \ldots, N$. The subscript $t$ in $x_t$ in $q(x_t|x_{1:t-1})$ indicates from which RV the samples are generated. We use $x_{1:t}^{(i)}$ as a shorthand notation for $(x_1^{(i)}, \ldots, x_t^{(i)})$. When $t = m$ we obtain that $x_{1:m}^{(i)} \sim q(x_{1:m})$. In other words, by sampling recursively $x_t^{(i)}$ from proposal distribution $q(x_t|x_{1:t-1})$ of RV with index $t$, we obtain a sample from $q(x_{1:m})$ at $t = m$. As is common in PF applications, we set $q(x_t|x_{1:t-1}) = p(x_t|x_{1:t-1})$, i.e., the proposal distribution is set to the conditional distribution of $p$.

We observe that the order of sampling follows the indexing of RVs with the index set $S$. However, there is not natural order of RVs on GRF, and the order of RV indices in $S$ does not have any particular meaning in that this order is not related in any way to our objective function $f$. The classical PF framework has been developed for sequential state estimation like tracking or robot localization [5], where observations arrive sequentially, and consequently, determine a natural order of RVs representing the states like locations. In a recent work [20], PF framework has been extended to work with unordered set of RVs for solving image jigsaw puzzles. Inspired by this work, we extend PF framework to solve $U - \mathcal{M}$ *maximal clique* problem in the weighted graph. Unlike tracking a moving object, in our problem, the observations are known from the beginning and are given by the affinity matrix $A$.

The key idea of [20] is to explore different orders of the states $(x_{i_1}, \ldots, x_{i_n})$ as opposed to utilizing the fix order of the states $x = (x_1, \ldots, x_n)$ determined by the index of RVs as in the standard PF. (States are assigned values of RVs.) To achieve this the first step of the PF algorithm is modified so that the importance sampling is performed for every RV not yet represented by the current particle. To formally define the sampling rule, we need to explicitly represent different orders of states with an index selection function $\sigma : \{1, \ldots, t\} \to \{1, \ldots, n\}$ for $1 < t \leq n$, which is one-to-one. In particular, when $t = n$, $\sigma$ is a permutation. We use the shorthand notation $\sigma(1 : t)$ to denote $(\sigma(1), \sigma(2), \ldots, \sigma(t))$ for $t \leq n$, and similarly, $x_{\sigma(1:t)} = (x_{\sigma(1)}, x_{\sigma(2)}, \ldots, x_{\sigma(t)})$. Each particle $x_{\sigma(1:t)}^{(i)}$ can now have a different permutation $\sigma^{(i)}$ representing the indices of RVs with assigned values. Thus, a sequence of RVs visited before time $t$ is described by a subsequence $(i_1, \ldots, i_t)$ of $t$ different numbers in $S = \{1, \ldots, n\}$.

We define an index set of indices of graph vertices that are compatible with selected vertices in $\sigma^{(i)}(1 : t)$ as $\kappa(\sigma^{(i)}(1 : t)) = S \setminus (\sigma^{(i)}(1 : t) \cup \mathrm{mutex}(\sigma^{(i)}(1 : t)))$. Hence $\kappa(\sigma^{(i)}(1 : t))$ contains indices from $S$ that that are both not present in $\sigma^{(i)}(1 : t)$ and not have mutex relation with the members of $\sigma^{(i)}(1 : t)$.

We are now ready to formulate the proposed importance sampling. At each iteration $t \leq n$, for each particle $(i)$ and for each $s \in \kappa(\sigma^{(i)}(1 : t-1))$, we sample $x_s^{(i)} \sim p(x_s|x_{\sigma(1:t-1)}^{(i)})$. The subscript $s$ at the conditional pdf $p$ indicates that we sample values for RV with index $s$. We generate at least one sample for each $s \in \kappa(\sigma^{(i)}(1 : t - 1))$. This means that the single particle $x_{\sigma(1:t-1)}^{(i)}$ is multiplied and extended to several follower particles $x_{\sigma(1:t-1),s}^{(i)}$.

Based on (2), it is easy to derive a formula for the proposal function:

$$p(x_s|x_{\sigma(1:t-1)}) = \frac{p(x_{\sigma(1:t-1)}, x_s)}{p(x_{\sigma(1:t-1)})} = \frac{\exp\frac{f(x_{\sigma(1:t-1)}, x_s)}{\gamma}}{\exp\frac{f(x_{\sigma(1:t-1)})}{\gamma}} = \exp\frac{f(x_{\sigma(1:t-1)}, x_s) - f(x_{\sigma(1:t-1)})}{\gamma}$$

(3)

We observe that $f(x_s, x_{\sigma(1:t-1)}) - f(x_{\sigma(1:t-1)}) = x_s^T A x_s + 2x_s^T A x_{\sigma(1:t-1)}$ is the gain in the target function $f$ obtained after assigning the value to RV $X_s$. Since we are interested in making this gain as large as possible, and assigning $x_s = 0$ leads to zero gain, we focus only on assigning $x_s = 1$. Consequently, the pdf in (3) can be treated as a probability mass function (pmf) over

$s \in \kappa(\sigma^{(i)}(1:t-1))$ and sampling from it becomes equivalent to sampling

$$s^{(i)} \sim p(s|\sigma^{(i)}(1:t-1)) = p(x_s = 1|x^{(i)}_{\sigma(1:t-1)}). \tag{4}$$

Hence, we can interpret a particle $x^{(i)}_{\sigma(1:t-1)}$ as a sequence of indices of selected graph vertices $\sigma^{(i)}(1:t-1)$, since $x^{(i)}_{\sigma(1:t-1)}$ is a vector of ones assigned to RVs with indices in $\sigma^{(i)}(1:t-1)$. In other words, it holds $\mathrm{ind}(x^{(i)}_{\sigma(1:t-1)}) = \sigma^{(i)}(1:t-1)$, where $\mathrm{ind} : \{0,1\}^n \to 2^S$ is a function that assigns to $x$ a set of indices of coordinates of $x$ that are equal to one. For example, if $x = (0,1,1,0,0) \in \{0,1\}^5$, then $\mathrm{ind}(x) = \{2,3\}$, which means that graph vertices with indices 2 and 3 are selected by $x$.

In order to construct the pmf in (4), we only need to assign the probabilities to all indices $s \in \kappa(\sigma^{(i)}(1:t-1))$ according to the definition in (3). Then $s^{(i)}$ is sampled from the discrete pmf constructed this way. Now we are ready to summarize the proposed PF framework in Algorithm 2.

---

**Algorithm 2** Particle Filter Algorithm for $U - \mathcal{M}$ Maximal Clique

---

**Input**: $A, U, \mathcal{M}, K, N, \gamma$.
**Initialize**: $t = 1$, initialize every particle $(i)$ with $\sigma_1^{(i)} = U$ for $i = 1, \ldots, N$.
**while** $\kappa(\sigma^{(1)}(1:t-1)) \cup \ldots \cup \kappa(\sigma^{(N)}(1:t-1)) \neq \emptyset$ and $t \leq K$ **do**
   **for** $i = 1 \to N$ **do**
     **if** $\kappa(\sigma^{(i)}(1:t-1)) \neq \emptyset$ **then**
       1. **Importance sampling / proposal:** Sample followers $x_s^{(i)}$ of particle $(i)$ from

$$x_s^{(i)} \sim p(x_s|x^{(i)}_{\sigma(1:t-1)}) = \exp((f(x_s, x^{(i)}_{\sigma(1:t-1)}) - f(x^{(i)}_{\sigma(1:t-1)}))/\gamma)$$

       and set $x^{(i,s)}_{\sigma(1:t)} = (x^{(i)}_{\sigma(1:t-1)}, x_s^{(i)})$ and $\sigma^{(i,s)}(t) = s$, i.e., $\sigma^{(i,s)}(1:t) = (\sigma(1:t-1), s)$.
       2. **Importance weighting / evaluation:** An individual importance weight is assigned to each follower particle according to

$$w(x^{(i,s)}_{\sigma(1:t)}) = \exp(f(x_s^{(i)}, x^{(i)}_{\sigma(1:t-1)})/\gamma)$$

     **else**
       we carry over the particle: $x^{(i,s)}_{\sigma(1:t)} = x^{(i)}_{\sigma(1:t-1)}$ and $w(x^{(i,s)}_{\sigma(1:t)}) = w(x^{(i)}_{\sigma(1:t-1)})$.
     **end if**
   **end for**
   3. **Resampling:** Sample with replacement $N$ new particle filters from $\{x^{(1,s)}_{\sigma(1:t)}, \ldots, x^{(N,s)}_{\sigma(1:t)}\}$ according to weights, and assign the sampled set to $\{x^{(1)}_{\sigma(1:t)}, \ldots, x^{(N)}_{\sigma(1:t)}\}$; set $t \leftarrow t+1$.
**end while**
**Output**: $\{x^{(1)}_{\sigma(1:t)}, \ldots, x^{(N)}_{\sigma(1:t)}\}$

---

We take the particle with maximal value of $f$ as solution of (2), or equivalently, as solution of (1): $x^* = x^{(k)}_{\sigma(1:t)}$, where $k = \arg\max_i f(x^{(i)}_{\sigma(1:t)})$. As proven in [20], $x^*$ approximates $\max_x p(x)$ with any precision for sufficiently large number of particles $N$.

## 5 Object Detection with the Deformable Part Model

In Section 3.2, we find $K^H$ salient parts denoted as $\mathcal{SP} = \{Q_i|i = 1, \ldots, K^H\}$ to represent an object class, each part $Q_i$ contains $K$ image patches, one patch from each training image. Now we describe the object model constructed from $\mathcal{SP}$.

We train a linear SVM classifier for each part $Q_i$, which we denote as $SVM(Q_i)$. To train the linear SVM classifier $SVM(Q_i)$, positive examples are the patches of $Q_i$. The negatives examples are obtained by an iterative procedure described in [10]. The initial training set consists of randomly chosen background windows and objects from other classes. The resulting classifier is used to scan images and select the top false positives as hard examples. These hard examples are added to the negative set and a new classifier is learned. This procedure is repeated several times to obtain the final classifier.

As in [6], we capture the spatial distribution of salient parts in $\mathcal{SP}$ with a star model, where the location of each part is expressed as an offset vector with respect to the model center. The offset is learned from the offsets of the patches in $Q_i$ to the centers of their training images (bounding boxes) containing them.

In order to be able to directly compare to Latent SVM [6], we use the same object detection framework. Thus, the detection is performed in the sliding window fashion followed by non maxima suppression. However, we do not use the root filter, which is an appearance classifier of the whole detection window. Thus, our detection is purely part based.

## 6 Experimental Evaluation

We validate our method on two datasets with deformable objects: ETHZ Giraffes dataset [9] and TUD-Pedestrians dataset [1]. For ETHZ Giraffes dataset, we follow the train/test split described in [18]: the first 43 giraffe images are positive training examples. The remaining 44 giraffe images in ETHZ dataset are used for testing as positive images. We also select 43 images from other categories as negative training images. As negative test images we take all remaining images from the other categories. Thus, we have the total of 86 training images, and the total of 169 test images. For learning the salient parts, the giraffe bounding boxes are normalized to the area of 3000 pixels with aspect ratio kept.

For TUD-Pedestrians dataset, we use the provided 400 images for training and 250 images for testing. The background of training images is used to extract negative examples. The training pedestrian bounding boxes are normalized to the height of 200 pixels with aspect ratio kept.

For both datasets, the size of each patch is $61 * 61$ pixels, number of patches per image is about 1000. We set $K^H$ in (C4) to 6 meaning that our goal is to learn 6 salient parts for each object class. The number of salient part was determined experimentally. The minimal distance $\tau$ between salient parts is 60 pixels for the giraffe class and 45 pixels for the pedestrian class. In Algorithm 2, the normalization parameter $\gamma$ is set to the median value in $A$ times the size of expected maximal clique times 2, the number of particles is $N = 500$, and for each particle we sample 10 followers. In order to compare to [6], we used the released latent SVM code [7] on the same training and testing images as for our approach.

### 6.1 Detection Performance

We plot the precision/recall (PR) curves to show the detection performance of the latent SVM method [6] and our method on both test dataset in Fig. 2. On the ETHZ giraffe class our average precision (AP) is 0.841, it is much better than AP of the latent SVM which is 0.610. Our result significantly outperforms the currently best reported result in [18], which has AP of 0.787. On the TUD-Pedestrian dataset, our AP of 0.862 is comparable to the latent SVM, whose AP is 0.875. These results show that our method can learn object models that yield very good detection performance. Our method is particularly suitable for learning part models of objects with large deformation like giraffes. The significant nonrigid deformation of giraffes leads to a large variation in the position of patches representing the same object part. Since latent SVM learning is based on incremental improvement in the position of parts, it seems to be unable to deal with large variations of part positions. In contrast, this does not influence the performance of our method, since it is matching based. Because the variance in the part positions in TUD-Pedestrian dataset is smaller than in giraffes, the performance of both methods becomes comparable. Some of our detection results are shown in Fig. 3. They demonstrate that our learned part model leads to detection performance that is robust to the scale changes, appearance variance, part location variance, and substantial occlusion.

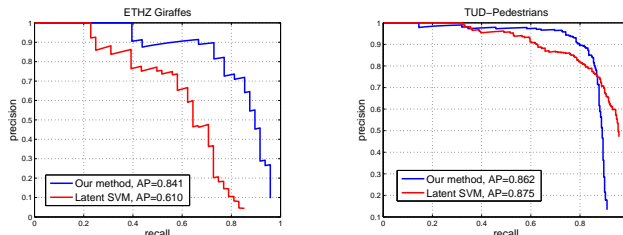

Figure 2: Precision/recall curves for Latent SVM method (red) and our method (blue) on ETHZ Giraffe dataset (left) and TUD-Pedestrian dataset (right).

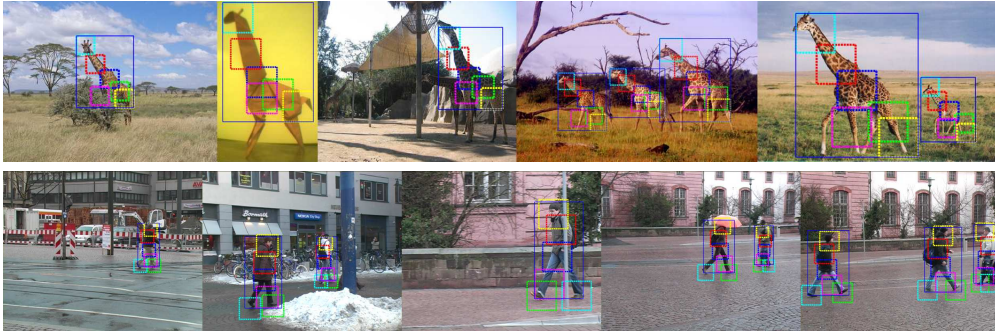

Figure 3: Some of our detection results for giraffe class and pedestrian dataset. The detected patches with the same color belong to the same salient part. The part colors are the same as in Fig. 4. Detected bounding boxes are shown in blue.

## 6.2 Tree Structure of Salient Parts

In our framework, it is also possible to learn a tree structure of the salient parts. Given a set of learned salient parts $\mathcal{SP} = \{Q_i | i = 1, \ldots, K^H\}$ as vertices, we construct a new graph, called Salient Part Graph (SPG). The edge weights of SPG are given by the average distance between pairs of salient parts $Q_i$ and $Q_j$ given by $D(i, j)$ for $i, j = 1, \ldots, K^H$.

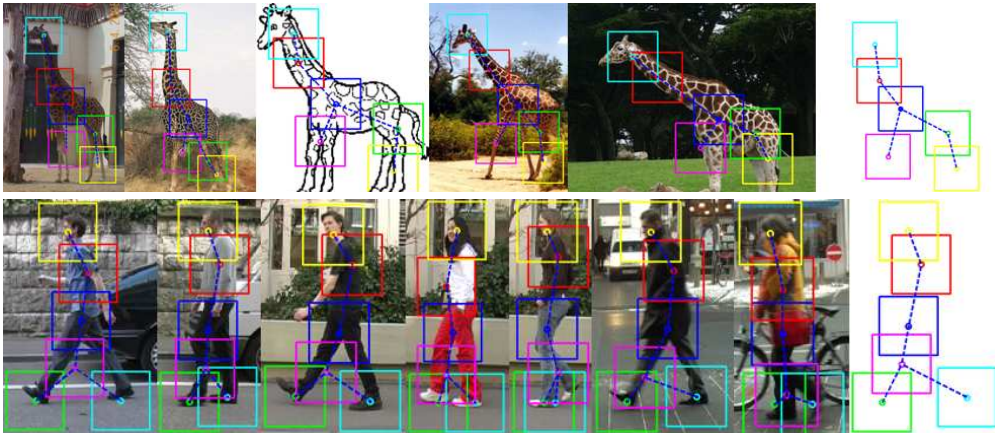

Figure 4: The learned salient parts and graph structures for the giraffe class and pedestrian dataset. The patches that belong to the same salient part are in the same color.

We obtain a minimum spanning tree of SPG using the Kruskal's algorithm [11]. The learned trees for two object classes of giraffes and pedestrians are illustrated in Fig. 4. Their connections yield a salient part structure in accord with our intuition. We did not utilize this tree structure for object detection. Instead we used the star model in our detection results in order to have a fair comparison to [6].

## 7 Conclusions

An object part is defined as a set of image patches. Learning object parts is formulated as two instances of the problem of finding maximal cliques in weighted graphs that satisfy hard constraints, and solved with the proposed Particle Filter inference framework. By utilizing the spatial relation of the obtained salient parts, we are also able to learn a tree structure of the deformable object model. The application of the proposed inference framework is not limited to learning object part models. There exist many other applications where it is important to enforce hard constraints like common pattern discovery and solving constrained matching problems.

**Acknowledgement**: The work was supported by the NSF under Grants IIS-0812118, BCS-0924164, OIA-1027897, by the AFOSR Grant FA9550-09-1-0207, and by the National Natural Science Foundation of China (NSFC) Grants 60903096, 61173120 and 60873127.

## Footnotes

*This work was done while the author visiting Temple University.

†This work was done when the author was a graduate student at Temple University.

# References

[1] M. Andriluka, S. Roth, and B. Schiele. People-tracking-by-detection and people-detection-by-tracking. *IEEE Conf. on Computer Vision and Pattern Recognition (CVPR)*, 2008.

[2] Y. Asahiro, R. Hassin, and K. Iwama. Complexity of finding dense subgraphs. *Discrete Applied Mathematics*, 121:15 – 26, 2002.

[3] D. Crisan and A. Doucet. A survey of convergence results on particle filtering methods for practitioners. *IEEE Transactions on Signal Processing*, 50(3):736–746, 2002.

[4] P. Dollar, B. Babenko, S. Belongie, P. Perona, and Z. Tu. Multiple component learning for object detection. *ECCV*, 2008.

[5] A. Eliazar and P. Ronald. Hierarchical linear/constant time slam using particle filters for dense maps. In *Advances in Neural Information Processing Systems 18*, pages 339–346. 2006.

[6] P. Felzenszwalb, R. Girshick, D. McAllester, and D. Ramanan. Object detection with discriminatively trained part based models. *IEEE Transactions on Pattern Analysis and Machine Intelligence*, Vol. 32, No. 9, 2010.

[7] P. F. Felzenszwalb, R. B. Girshick, and D. McAllester. Discriminatively trained deformable part models, release 4. http://people.cs.uchicago.edu/ pff/latent-release4/.

[8] R. Fergus, P. Perona, and A. Zisserman. Object class recognition by unsupervised scale-invariant learning. *Proc. of the IEEE Conf on Computer Vision and Pattern Recognition*, 2003.

[9] V. Ferrari, T. Tuytelaars, and L. V. Gool. Object detection by contour segment networks. *ECCV*, 2006.

[10] H. Harzallah, F. Jurie, and C. Schmid. Combining efficient object localization and image classification. In *International Conference on Computer Vision*, 2009.

[11] J. B. Kruskal. On the shortest spanning subtree of a graph and the traveling salesman problem. In *Proceedings of the American Mathematical Society*, 1956.

[12] M. Leordeanu, M. Hebert, and R. Sukthankar. An integer projected fixed point method for graph matching and map inference. In *Neural Info. Proc. Systems (NIPS)*, 2009.

[13] Z. Lin, G. Hua, and L. S. Davis. Multiple instance feature for robust part-based object detection. *IEEE Conference on Computer Vision and Pattern Recognition*, 2009.

[14] H. Liu, L. J. Latecki, and S. Yan. Robust clustering as ensemble of affinity relations. In *Neural Info. Proc. Systems (NIPS)*, 2010.

[15] N. Loeff, H. Arora, A. Sorokin, and D. Forsyth. Efficient unsupervised learning for localization and detection in object categories. In *Advances in Neural Information Processing Systems 18*, pages 811–818. 2006.

[16] M. Pavan and M. Pelillo. Dominant sets and pairwise clustering. *IEEE Transactions on Pattern Analysis and Machine Intelligence*, 29:167-172, 2007.

[17] A. Quattoni, M. Collins, and T. Darrell. Conditional random fields for object recognition. In *Advances in Neural Information Processing Systems 17*, pages 1097–1104. 2005.

[18] P. Srinivasan, Q. Zhu, and J. Shi. Many-to-one contour matching for describing and discriminating object shape. *IEEE Conference on Computer Vision and Pattern Recognition*, 2010.

[19] X. Wang, X. Bai, W. Liu, and L. J. Latecki. Feature context for image classification and object detection. *IEEE Conf. on Computer Vision and Pattern Recognition (CVPR)*, 2011.

[20] X. Yang, N. Adluru, and L. J. Latecki. Particle filter with state permutations for solving image jigsaw puzzles. In *IEEE Conf. on Computer Vision and Pattern Recognition (CVPR)*, 2011.

